# A Natural Policy Gradient

**Sham Kakade**
Gatsby Computational Neuroscience Unit
17 Queen Square, London, UK WC1N 3AR
http://www.gatsby.ucl.ac.uk
*sham@gatsby.ucl.ac.uk*

## Abstract

We provide a natural gradient method that represents the steepest descent direction based on the underlying structure of the parameter space. Although gradient methods cannot make large changes in the values of the parameters, we show that the natural gradient is moving toward choosing a greedy optimal action rather than just a better action. These greedy optimal actions are those that would be chosen under one improvement step of policy iteration with approximate, *compatible* value functions, as defined by Sutton *et al.* [9]. We then show drastic performance improvements in simple MDPs and in the more challenging MDP of Tetris.

## 1 Introduction

There has been a growing interest in direct policy-gradient methods for approximate planning in large Markov decision problems (MDPs). Such methods seek to find a good policy $\pi$ among some restricted class of policies, by following the gradient of the future reward. Unfortunately, the standard gradient descent rule is non-covariant. Crudely speaking, the rule $\Delta\theta_i = \alpha\partial f/\partial\theta_i$ is dimensionally inconsistent since the left hand side has units of $\theta_i$ and the right hand side has units of $1/\theta_i$ (and all $\theta_i$ do not necessarily have the same dimensions).

In this paper, we present a covariant gradient by defining a metric based on the underlying structure of the policy. We make the connection to policy iteration by showing that the natural gradient is moving toward choosing a greedy optimal action. We then analyze the performance of the natural gradient in both simple and complicated MDPs. Consistent with Amari's findings [1], our work suggests that the plateau phenomenon might not be as severe using this method.

## 2 A Natural Gradient

A finite MDP is a tuple $(S, s_0, A, R, P)$ where: $S$ is finite set of states, $s_0$ is a start state, $A$ is a finite set of actions, $R$ is a reward function $R : S \times A \to [0, R_{max}]$, and $P$ is the transition model. The agent's decision making procedure is characterized by a stochastic policy $\pi(a; s)$, which is the probability of taking action $a$ in state $s$ (a semi-colon is used to distinguish the random variables from the parameters of

the distribution). We make the assumption that *every policy $\pi$ is ergodic, ie* has a well-defined stationary distribution $\rho^\pi$. Under this assumption, the *average reward* (or undiscounted reward) is $\eta(\pi) \equiv \sum_{s,a} \rho^\pi(s)\pi(a;s)R(s,a)$, the state-action value is $Q^\pi(s,a) \equiv E_\pi\{\sum_{t=0}^\infty R(s_t,a_t) - \eta(\pi)|s_0 = s, a_0 = a\}$ and the value function is $J^\pi(s) \equiv E_{\pi(a';s)}\{Q^\pi(s,a')\}$, where and $s_t$ and $a_t$ are the state and action at time $t$. We consider the more difficult case where the goal of the agent is to find a policy that maximizes the average reward over some restricted class of smoothly parameterized policies, $\tilde{\Pi} = \{\pi_\theta : \theta \in \Re^m\}$, where $\pi_\theta$ represents the policy $\pi(a;s,\theta)$.

The exact gradient of the average reward (see [8, 9]) is:

$$\nabla\eta(\theta) = \sum_{s,a} \rho^\pi(s)\nabla\pi(a;s,\theta)Q^\pi(s,a) \tag{1}$$

where we abuse notation by using $\eta(\theta)$ instead of $\eta(\pi_\theta)$. The steepest descent direction of $\eta(\theta)$ is defined as the vector $d\theta$ that minimizes $\eta(\theta + d\theta)$ under the constraint that the squared length $|d\theta|^2$ is held to a small constant. This squared length is defined with respect to some positive-definite matrix $G(\theta)$, *ie* $|d\theta|^2 \equiv \sum_{ij} G_{ij}(\theta)d\theta_i d\theta_j = d\theta^T G(\theta)d\theta$ (using vector notation). The steepest descent direction is then given by $G^{-1}\nabla\eta(\theta)$ [1]. Standard gradient descent follows the direction $\nabla\eta(\theta)$ which is the steepest descent under the assumption that $G(\theta)$ is the identity matrix, $I$. However, this *as hoc* choice of a metric is not necessarily appropriate. As suggested by Amari [1], it is better to define a metric based not on the choice of coordinates but rather on the manifold (*ie* the surface) that these coordinates parameterize. This metric defines the natural gradient.

Though we slightly abuse notation by writing $\eta(\theta)$, the average reward is technically a function on the set of distributions $\{\pi_\theta : \theta \in \Re^m\}$. To each state $s$, there corresponds a probability manifold, where the distribution $\pi(a;s,\theta)$ is a point on this manifold with coordinates $\theta$. The Fisher information matrix of this distribution $\pi(a;s,\theta)$ is

$$F_s(\theta) \equiv E_{\pi(a;s,\theta)}\left[\frac{\partial \log \pi(a;s,\theta)}{\partial \theta_i} \frac{\partial \log \pi(a;s,\theta)}{\partial \theta_j}\right], \tag{2}$$

and it is clearly positive definite. As shown by Amari (see [1]), the Fisher information matrix, up to a scale, is an invariant metric on the space of the parameters of probability distributions. It is invariant in the sense that it defines the *same* distance between two points regardless of the choice of coordinates (*ie* the parameterization) used, unlike $G = I$.

Since the average reward is defined on a set of these distributions, the straightforward choice we make for the metric is:

$$F(\theta) \equiv E_{\rho^\pi(s)}[F_s(\theta)] \tag{3}$$

where the expectation is with respect to the stationary distribution of $\pi_\theta$. Notice that although each $F_s$ is independent of the parameters of the MDP's transition model, the weighting by the stationary distribution introduces dependence on these parameters. Intuitively, $F_s(\theta)$ measures distance on a probability manifold corresponding to state $s$ and $F(\theta)$ is the average such distance. The steepest descent direction this gives is:

$$\tilde{\nabla}\eta(\theta) \equiv F(\theta)^{-1}\nabla\eta(\theta). \tag{4}$$

# 3 The Natural Gradient and Policy Iteration

We now compare policy improvement under the natural gradient to policy iteration. For an appropriate comparison, consider the case in which $Q^\pi(s, a)$ is approximated by some *compatible* function approximator $f^\pi(s, a; \omega)$ parameterized by $\omega$ [9, 6].

## 3.1 Compatible Function Approximation

For vectors $\theta, \omega \in \Re^m$, we define:

$$\psi(s, a)^\pi = \nabla \log \pi(a; s, \theta), \quad f^\pi(s, a; \omega) = \omega^T \psi^\pi(s, a) \qquad (5)$$

where $[\nabla \log \pi(a; s, \theta)]_i = \partial \log \pi(a; s, \theta)/\partial \theta_i$. Let $\tilde{\omega}$ minimize the squared error $\epsilon(\omega, \pi) \equiv \sum_{s,a} \rho^\pi(s) \pi(a; s, \theta)(f^\pi(s, a; \omega) - Q^\pi(s, a))^2$. This function approximator is compatible with the policy in the sense that if we use the approximations $f^\pi(s, a; \tilde{\omega})$ in lieu of their true values to compute the gradient (equation 1), then the result would still be exact [9, 6] (and is thus a sensible choice to use in actor-critic schemes).

**Theorem 1.** *Let $\tilde{\omega}$ minimize the squared error $\epsilon(\omega, \pi_\theta)$. Then*

$$\tilde{\omega} = \tilde{\nabla} \eta(\theta).$$

*Proof.* Since $\tilde{\omega}$ minimizes the squared error, it satisfies the condition $\partial \epsilon / \partial \omega_i = 0$, which implies:

$$\sum_{s,a} \rho^\pi(s) \pi(a; s, \theta) \psi^\pi(s, a)(\psi^\pi(s, a)^T \tilde{\omega} - Q^\pi(s, a)) = 0.$$

or equivalently:

$$\left(\sum_{s,a} \rho^\pi(s) \pi(a; s, \theta) \psi^\pi(s, a) \psi^\pi(s, a)^T\right) \tilde{\omega} = \sum_{s,a} \rho^\pi(s) \pi(a; s, \theta) \psi^\pi(s, a) Q^\pi(s, a).$$

By definition of $\psi^\pi$, $\nabla \pi(a; s, \theta) = \pi(a; s, \theta) \psi^\pi(s, a)$ and so the right hand side is equal to $\nabla \eta$. Also by definition of $\psi^\pi$, $F(\theta) = \sum_{s,a} \rho^\pi(s) \pi(a; s, \theta) \psi^\pi(s, a) \psi^\pi(s, a)^T$. Substitution leads to:

$$F(\theta)\tilde{\omega} = \nabla \eta(\theta).$$

Solving for $\tilde{\omega}$ gives $\tilde{\omega} = F(\theta)^{-1} \nabla \eta(\theta)$, and the result follows from the definition of the natural gradient. □

Thus, sensible actor-critic frameworks (those using $f^\pi(s, a; \omega)$) are forced to use the natural gradient as the weights of a linear function approximator. If the function approximation is accurate, then good actions (*ie* those with large state-action values) have feature vectors that have a large inner product with the natural gradient.

## 3.2 Greedy Policy Improvement

A greedy policy improvement step using our function approximator would choose action $a$ in state $s$ if $a \in \text{argmax}_{a'} f^\pi(s, a'; \tilde{\omega})$. In this section, we show that the natural gradient tends to move toward this *best* action, rather than just a good action.

Let us first consider policies in the exponential family ($\pi(a; s, \theta) \propto \exp(\theta^T \phi_{sa})$ where $\phi_{sa}$ is some feature vector in $\Re^m$). The motivation for the exponential family is because it has affine geometry (*ie* the flat geometry of a plane), so a translation of a point by a tangent vector will keep the point on the manifold. In general, crudely

speaking, the probability manifold of $\pi(a; s, \theta)$ could be curved, so a translation of a point by a tangent vector would not necessarily keep the point on the manifold (such as on a sphere). We consider the general (non-exponential) case later.

We now show, for the exponential family, that a sufficiently large step in the natural gradient direction will lead to a policy that is equivalent to a policy found after a greedy policy improvement step.

**Theorem 2.** *For $\pi(a; s, \theta) \propto \exp(\theta^T \phi_{sa})$, assume that $\widetilde{\nabla}\eta(\theta)$ is non-zero and that $\tilde{\omega}$ minimizes the approximation error. Let $\pi_\infty(a; s) = \lim_{\alpha \to \infty} \pi(a; s, \theta + \alpha \widetilde{\nabla}\eta(\theta))$. Then $\pi_\infty(a; s) \neq 0$ if and only if $a \in \mathrm{argmax}_{a'} f^\pi(s, a'; \tilde{\omega})$.*

*Proof.* By the previous result, $f^\pi(s, a; \tilde{\omega}) = \widetilde{\nabla}\eta(\theta)^T \psi^\pi(s, a)$. By definition of $\pi(a; s, \theta)$, $\psi^\pi(s, a) = \phi_{sa} - E_{\pi(a'; s, \theta)}(\phi_{sa'})$. Since $E_{\pi(a'; s, \theta)}(\phi_{sa'})$ is not a function of $a$, it follows that

$$\mathrm{argmax}_{a'} f^\pi(s, a'; \tilde{\omega}) = \mathrm{argmax}_{a'} \widetilde{\nabla}\eta(\theta)^T \phi_{sa'} .$$

After a gradient step, $\pi(a; s, \theta + \alpha\widetilde{\nabla}\eta(\theta)) \propto \exp(\theta^T \phi_{sa} + \alpha\widetilde{\nabla}\eta(\theta)^T \phi_{sa})$. Since $\widetilde{\nabla}\eta(\theta) \neq 0$, it is clear that as $\alpha \to \infty$ the term $\widetilde{\nabla}\eta(\theta)^T \phi_{sa}$ dominates, and so $\pi_\infty(a, s) = 0$ if and only if $a \notin \mathrm{argmax}_{a'} \widetilde{\nabla}\eta(\theta)^T \phi_{sa'}$. □

It is in this sense that the natural gradient tends to move toward choosing the *best* action. It is straightforward to show that if the standard non-covariant gradient rule is used instead then $\pi_\infty(a; s)$ will select only a *better* action (not necessarily the best), *ie* it will choose an action $a$ such that $f^\pi(s, a; \tilde{\omega}) > E_{\pi(a'; s)}\{f^\pi(s, a'; \tilde{\omega})\}$. Our use of the exponential family was only to demonstrate this point in the extreme case of an infinite learning rate.

Let us return to case of a general parameterized policy. The following theorem shows that the natural gradient is locally moving toward the best action, determined by the local linear approximator for $Q^\pi(s, a)$.

**Theorem 3.** *Assume that $\tilde{\omega}$ minimizes the approximation error and let the update to the parameter be $\theta' = \theta + \alpha\widetilde{\nabla}\eta(\theta)$. Then*

$$\pi(a; s, \theta') = \pi(a; s, \theta)(1 + f^\pi(s, a; \tilde{\omega})) + O(\alpha^2)$$

*Proof.* The change in $\theta$, $\Delta\theta$, is $\alpha\widetilde{\nabla}\eta(\theta)$, so by theorem 1, $\Delta\theta = \alpha\tilde{\omega}$. To first order,

$$
\begin{aligned}
\pi(a; s, \theta') &= \pi(a; s, \theta) + \frac{\partial \pi(a; s, \theta)^T}{\partial \theta}\Delta\theta + O(\Delta\theta^2) \\
&= \pi(a; s, \theta)(1 + \psi(s, a)^T \Delta\theta) + O(\Delta\theta^2) \\
&= \pi(a; s, \theta)(1 + \alpha\psi(s, a)^T \tilde{\omega}) + O(\alpha^2) \\
&= \pi(a; s, \theta)(1 + \alpha f^\pi(s, a; \tilde{\omega})) + O(\alpha^2) ,
\end{aligned}
$$

where we have used the definition of $\psi$ and $f$. □

It is interesting to note that choosing the greedy action will not in general improve the policy, and many detailed studies have gone into understanding this failure [3]. However, with the overhead of a line search, we can guarantee improvement and move toward this greedy one step improvement. Initial improvement is guaranteed since $F$ is positive definite.

## 4   Metrics and Curvatures

Obviously, our choice of $F$ is not unique and the question arises as to whether or not there is a better metric to use than $F$. In the different setting of parameter estimation, the Fisher information converges to the Hessian, so it is asymptotically efficient [1], *ie* attains the Cramer-Rao bound. Our situation is more similar to the blind source separation case where a metric is chosen based on the underlying parameter space [1] (of non-singular matrices) and is not necessarily asymptotically efficient (*ie* does not attain second order convergence). As argued by Mackay [7], one strategy is to pull a metric out of the data-independent terms of the Hessian (if possible), and in fact, Mackay [7] arrives at the same result as Amari for the blind source separation case.

Although the previous sections argued that our choice is appropriate, we would like to understand how $F$ relates to the Hessian $\nabla^2 \eta(\theta)$, which, as shown in [5], has the form:

$$\nabla^2 \eta(\theta) = \sum_{sa} \rho^\pi(s)(\nabla^2\pi(a;s)Q^\pi(s,a)+\nabla\pi(a;s)\nabla Q^\pi(s,a)^T+\nabla Q^\pi(s,a)\nabla\pi(a;s)^T).$$

(6)

Unfortunately, all terms in this Hessian are data-dependent (*ie* are coupled to state-action values). It is clear that $F$ does not capture any information from these last two terms, due to their $\nabla Q^\pi$ dependence. The first term might have some relation to $F$ due the factor of $\nabla^2\pi$. However, the Q values weight this curvature of our policy and our metric is neglecting such weighting.

Similar to the blind source separation case, our metric clearly does not necessarily converge to the Hessian and so it is not necessarily asymptotically efficient (*ie* does not attain a second order convergence rate). However, in general, the Hessian will not be positive definite and so the curvature it provides could be of little use until $\theta$ is close to a local maxima. Conjugate methods would be expected to be more efficient near a local maximum.

## 5   Experiments

We first look at the performance of the natural gradient in a few simple MDPs before examining its performance in the more challenging MDP of Tetris. It is straightforward to estimate $F$ in an online manner, since the derivatives $\nabla \log \pi$ must be computed anyway to estimate $\nabla\eta(\theta)$. If the update rule

$$f \leftarrow f + \nabla \log \pi(a_t; s_t, \theta)\nabla \log \pi(a_t; s_t, \theta)^T$$

is used in a $T$-length trajectory, then $f/T$ is a consistent estimate of $F$. In our first two examples, we do not concern ourselves with sampling issues and instead numerically integrate the exact derivative ($\theta_t = \theta_0 + \int_0^t \nabla\eta(\theta_t)d\theta$). In all of our simulations, the policies tend to become deterministic ($\nabla \log \pi \to 0$) and to prevent $F$ from becoming singular, we add about $10^{-3}I$ at every step in all our simulations.

We simulated the natural policy gradient in a simple 1-dimensional linear quadratic regulator with dynamics $x(t + 1) = .7x(t) + u(t) + \epsilon(t)$ and noise distribution $\epsilon \sim G(0, 1)$. The goal is to apply a control signal $u$ to keep the system at $x = 0$, (incurring a cost of $x(t)^2$ at each step). The parameterized policy used was $\pi(u; x, \theta) \propto \exp(\theta_1 x^2 + \theta_2 x)$. Figure 1A shows the performance improvement when the units of the parameters are scaled by a factor of 10 (see figure text). Notice that the time to obtain a score of about 22 is about three orders of magnitude

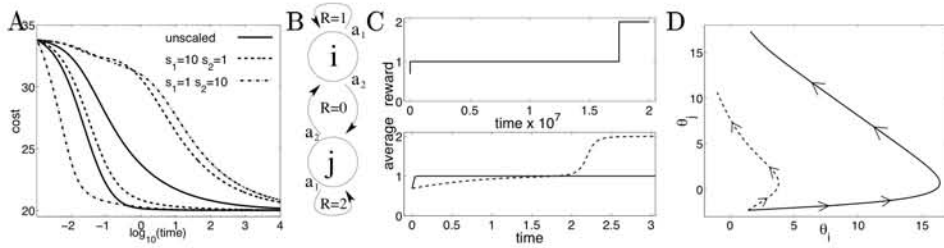

Figure 1: A) The cost vs. $\log_{10}(\text{time})$ for an LQG (with 20 time step trajectories). The policy used was $\pi(u; x, \theta) \propto \exp(\theta_1 s_1 x^2 + \theta_2 s_2 x)$ where the rescaling constants, $s_1$ and $s_2$, are shown in the legend. Under equivalent starting distributions ($\theta_1 s_1 = \theta_2 s_2 = -.8$), the right-most three curves are generated using the standard gradient method and the rest use the natural gradient. B) See text. C top) The average reward vs. time (on a $10^7$ scale) of a policy under standard gradient descent using the sigmoidal policy parameterization ($\pi(1; s, \theta_i) \propto \exp(\theta_i)/(1 + \exp(\theta_i))$, with the initial conditions $\pi(i, 1) = .8$ and $\pi(j, 1) = .1$. C bottom) The average reward vs. time (unscaled) under standard gradient descent (solid line) and natural gradient descent (dashed line) for an early window of the above plot. D) Phase space plot for the standard gradient case (the solid line) and the natural gradient case (dashed line).

faster. Also notice that the curves under different rescaling are *not identical*. This is because $F$ *is not an invariant metric* due to the weighting by $\rho_s$.

The effects of the weighting by $\rho(s)$ are particularly clear in a simple 2-state MDP (Figure 1B), which has self- and cross-transition actions and rewards as shown. Increasing the chance of a self-loop at $i$ decreases the stationary probability of $j$. Using a sigmoidal policy parameterization (see figure text) and initial conditions corresponding to $\rho(i) = .8$ and $\rho(j) = .2$, both self-loop action probabilities will initially be increased under a gradient rule (since one step policy improvement chooses the self-loop for each state). Since the standard gradient weights the learning to each parameter by $\rho(s)$ (see equation 1), the self-loop action at state $i$ is increased faster than the self loop probability at $j$, which has the effect of decreasing the effective learning-rate to state $j$ even further. This leads to an extremely flat plateau with average reward 1 (shown in Figure 1C top), where the learning for state $j$ is thwarted by its low stationary probability. This problem is so severe that before the optimal policy is reached $\rho(j)$ drops as low as $10^{-7}$ from its initial value of .2, which is disastrous for sampling methods. Figure 1C bottom shows the performance of the natural gradient (in a very early time window of Figure 1C top). Not only is the time to the optimal policy decreased by a factor of $10^7$, the stationary distribution of state $i$ never drops below .05. Note though the standard gradient does increase the average reward faster at the start, but only to be seduced by sticking at state $i$. The phase space plot in Figure 1D shows the uneven learning to the different parameters, which is at the heart of the problem. In general, if a table lookup Boltzmann policy is used (ie $\pi(a; s, \theta) \propto exp(\theta_{sa})$), it is straightforward to show that the natural gradient weights the components of $\tilde{\nabla}\eta$ uniformly (instead of using $\rho(s)$), thus evening evening out the learning to all parameters.

The game of Tetris provides a challenging high dimensional problem. As shown in [3], greedy policy iteration methods using a linear function approximator exhibit drastic performance degradation after providing impressive improvement (see [3] for a description of the game, methods, and results). The upper curve in Figure2A replicates these results. Tetris provides an interesting case to test gradient methods,

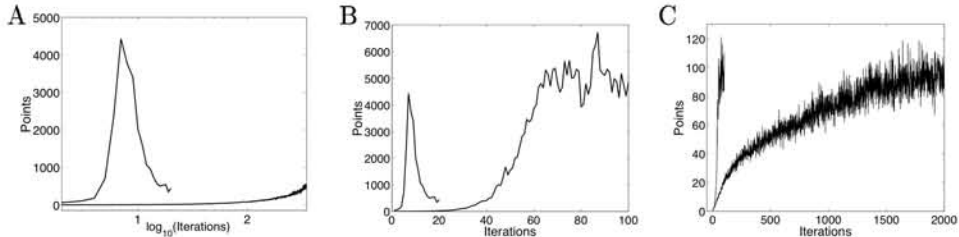

Figure 2: A) Points vs. log(Iterations). The top curve duplicates the same results in [3] using the same features (which were simple functions of the heights of each column and the number of holes in the game). We have no explanation for this performance degradation (nor does [3]). The lower curve shows the poor performance of the standard gradient rule. B) The curve on the right shows the natural policy gradient method (and uses the biased gradient method of [2] though this method alone gave poor performance). We found we could obtain faster improvement and higher asymptotes if the robustifying factor of $10^{-3}I$ that we added to $F$ was more carefully controlled (we did not carefully control the parameters). C) Due to the intensive computational power required of these simulations we ran the gradient in a smaller Tetris game (height of 10 rather than 20) to demonstrate that the standard gradient updates (right curve) would eventually reach the same performance of the natural gradient (left curve).

which are guaranteed not to degrade the policy. We consider a policy compatible with the linear function approximator used in [3] (*ie* $\pi(a; s, \theta) \propto \exp(\theta^T \phi_{sa})$ where $\phi_{sa}$ are the same feature vectors). The features used in [3] are the heights of each column, the differences in height between adjacent columns, the maximum height, and the number of 'holes'. The lower curve in Figure 2A shows the particularly poor performance of the standard gradient method. In an attempt to speed learning, we tried a variety of more sophisticated methods to no avail, such as conjugate methods, weight decay, annealing, the variance reduction method of [2], the Hessian in equation 6, *etc*. Figure 2B shows a drastic improvement using the natural gradient (note that the timescale is linear). This performance is consistent with our theoretical results in section 3, which showed that the natural gradient is moving toward the solution of a greedy policy improvement step. The performance is somewhat slower than the greedy policy iteration (left curve in Figure 2B) which is to be expected using smaller steps. However, the policy does not degrade with a gradient method. Figure 2 shows that the performance of the standard gradient rule (right curve) eventually reaches the the same performance of the natural gradient, in a scaled down version of the game (see figure text).

## 6   Discussion

Although gradient methods cannot make large policy changes compared to greedy policy iteration, section 3 implies that these two methods might not be that disparate, since a natural gradient method is moving toward the solution of a policy improvement step. With the overhead of a line search, the methods are even more similar. The benefit is that performance improvement is now guaranteed, unlike in a greedy policy iteration step.

It is interesting, and unfortunate, to note that the $F$ does not asymptotically converge to the Hessian, so conjugate gradient methods might be more sensible asymptotically. However, far from the converge point, the Hessian is not necessarily

informative, and the natural gradient could be more efficient (as demonstrated in Tetris). The intuition as to why the natural gradient could be efficient far from the maximum, is that it is pushing the policy toward choosing greedy optimal actions. Often, the region (in parameter space) far from from the maximum is where large performance changes could occur. Sufficiently close to the maximum, little performance change occurs (due to the small gradient), so although conjugate methods might converge faster near the maximum, the corresponding performance change might be negligible. More experimental work is necessary to further understand the effectiveness of the natural gradient.

## Acknowledgments

We thank Emo Todorov and Peter Dayan for many helpful discussions. Funding is from the NSF and the Gatsby Charitable Foundation.

## References

[1] S. Amari. Natural gradient works efficiently in learning. *Neural Computation*, 10(2):251–276, 1998.

[2] J. Baxter and P. Bartlett. Direct gradient-based reinforcement learning. Technical report, Australian National University, Research School of Information Sciences and Engineering, July 1999.

[3] D. P. Bertsekas and J. N. Tsitsiklis. *Neuro-Dynamic Programming*. Athena Scientific, 1996.

[4] P. Dayan and G. Hinton. Using em for reinforcement learning. *Neural Computation*, 9:271–278, 1997.

[5] S. Kakade. Optimizing average reward using discounted reward. *COLT. in press.*, 2001.

[6] V. Konda and J. Tsitsiklis. Actor-critic algorithms. *Advances in Neural Information Processing Systems*, 12, 2000.

[7] D. MacKay. Maximum likelihood and covariant algorithms for independent component analysis. Technical report, University of Cambridge, 1996.

[8] P. Marbach and J. Tsitsiklis. Simulation-based optimization of markov reward processes. Technical report, Massachusetts Institute of Technology, 1998.

[9] R. Sutton, D. McAllester, S. Singh, and Y. Mansour. Policy gradient methods for reinforcement learning with function approximation. *Neural Information Processing Systems*, 13, 2000.

[10] L. Xu and M. I. Jordan. On convergence properties of the EM algorithm for gaussian mixtures. *Neural Computation*, 8(1):129–151, 1996.
